# Differentially Private M-Estimators

**Lei, Jing**
Department of Statistics
Carnegie Mellon University
Pittsburgh, PA 15213
jinglei@andrew.cmu.edu

## Abstract

This paper studies privacy preserving M-estimators using perturbed histograms. The proposed approach allows the release of a wide class of M-estimators with both differential privacy and statistical utility without knowing *a priori* the particular inference procedure. The performance of the proposed method is demonstrated through a careful study of the convergence rates. A practical algorithm is given and applied on a real world data set containing both continuous and categorical variables.

## 1 Introduction

Privacy-preserving data analysis has received increasing attention in recent years. Among various notions of privacy, *differential privacy* [1, 2] provides mathematically rigorous privacy guarantee and protects against essentially all kinds of identity attacks regardless of the auxiliary information that may be available to the attackers. Differential privacy requires that the presence or absence of any individual data record can never greatly change the outcome and hence the user can hardly learn much about any individual data record from the output.

However, designing differentially private statistical inference procedures has been a challenging problem. Differential privacy protects individual data by introducing uncertainty in the outcome, which generally requires the output of any inference procedure to be random even for a fixed input data set. This makes differentially private statistical analysis different from most traditional statistical inference procedures, which are deterministic once the data set is given. Most existing works [3, 4, 5] focus on the interactive data release where a particular statistical inference problem is chosen *a priori* and the randomized output for that particular inference is released to the users. In reality a data release that allows multiple inference procedures are often desired because real world statistical analyses usually consist of a series of inferences such as exploratory analysis, model fitting, and model selection, where the exact inference problem in a later stage is determined by results of previous steps and cannot be determined in advance.

In this work we study M-estimators under a differentially private framework. The proposed method uses perturbed histograms to provide a systematic way of releasing a class of M-estimators in a non-interactive fashion. Such a non-interactive method uses randomization independent of any particular inference procedure, therefore it allows the users to apply different inference procedures on the same synthetic data set without additional privacy compromise. The accuracy of these private preserving estimates has also been studied and we prove that, under mild conditions on the contrast functions of the M-estimators, the proposed differentially private M-estimators are consistent. As a special case, this approach gives $1/\sqrt{n}$-consistent estimates for quantiles, providing a simple and efficient alternative solution to similar problems considered in [4, 5]. Our main condition requires convexity and bounded partial derivatives of the contrast function. The convexity is used to ensure the existence and stability of the M-estimator whereas the bounded derivative controls the bias caused by the perturbed histogram. In classical theory of M-estimators, a contrast function with

bounded derivative implies robustness of the corresponding M-estimator. This is another evidence of the natural connection between robustness and differential privacy [4].

We also describe an algorithm that is conceptually simple and computationally feasible. It is flexible enough to accommodate continuous, ordinal, and categorical variables at the same time, as demonstrated by its application on a Bay Area housing data.

## 1.1 Related Work

The perturbed histogram is first described under the context of differential privacy in [1]. The problem of non-interactive release has also been studied by [6], which targets at releasing the differentially private distribution function or the density function in a non-parametric setting. Theoretically, M-estimators could be indirectly obtained from the released density function. However, the more direct perspective taken in this paper leads to an improved rate of convergence as well as an efficient algorithm.

Several aspects of parameter estimation problems have been studied with differential privacy under the interactive framework. In particular, [4] shows that many robust estimators can be made differentially private and that general private estimators can be obtained from composition of robust location and scale estimators. [5] shows that statistical estimators with generic asymptotic normality can be made differentially private with the same asymptotic variance. Both works involve estimating the inter-quartile range in a differentially private manner, where the algorithm may output "No Response" [4], or the data is assumed to have known upper and lower bounds [5]. In a slightly different context, [3] considers penalized logistic regression as a special case of empirical risk minimization, where the penalized logistic regression coefficients are estimated with differential privacy by minimizing a perturbed objective function. Their method uses a different form of perturbation and is still interactive. It connects with the present paper in the sense that the perturbation is finally expressed in the objective function. Both papers assume convexity, which ensures that the shift in the minimizer is small when the deviation in the objective function is small. We also note that the method in [3] depends on a strictly convex penalty term which is typically used in high-dimensional problems, while our method works for problems where no penalization is used.

## 2 Preliminaries

### 2.1 Definition of Privacy

A database is modeled as a set of data points $D = \{x_1, \ldots, x_n\} \in \mathcal{X}^n$, where $\mathcal{X}$ is the data universe. In most cases each data entry $x_i$ represents the microdata of an individual. We use the Hamming distance to measure the proximity between two databases of the same size. Suppose $|D| = |D'|$, the Hamming distance is $H(D, D') = |D \backslash D'| = |D' \backslash D|$. The objective of data privacy is to release useful information from the data set while protecting information about any individual data entry.

**Definition 1** (Differential Privacy [1]). A randomized function $T(D)$ gives $\alpha$-differential privacy if for all pairs of databases $(D, D')$ with $H(D, D') = 1$ and all measurable subsets $E$ of the image of $T$:

$$\left| \log \frac{P(T \in E | D)}{P(T \in E | D')} \right| \leq \alpha. \tag{1}$$

In the rest of this paper we assume that, $n$, the size of database, is public.

### 2.2 The Perturbed Histogram

In most statistical problems, a database $D$ consists of $n$ independent copies of a random variable $X$ with density $f(x)$. For simplicity, we assume $\mathcal{X} = [0,1]^d$. As we will see in Section 3.2, our method can be extended to non-compact $\mathcal{X}$ for some important examples. Suppose $[0,1]^d$ is partitioned into cubic cells with equal bandwidth $h_n$ such that $k_n = h_n^{-1}$ is an integer. Denote each cell as $B_{\mathbf{r}} = \bigotimes_{j=1}^d [(r_j - 1)h_n, r_j h_n),^1$ for all $\mathbf{r} = (r_1, ..., r_d) \in \{1, ..., k_n\}^d$. The histogram

density estimator is then

$$\hat{f}_{\text{hist}}(x) = h_n^{-d} \sum_{\mathbf{r}} \frac{n_{\mathbf{r}}}{n} \mathbf{1}(x \in B_{\mathbf{r}}), \tag{2}$$

where $n_{\mathbf{r}} := \sum_{i=1}^n \mathbf{1}(X_i \in B_{\mathbf{r}})$ is the number of data points in $B_{\mathbf{r}}$.

Clearly the density estimator described above depends on the data only through the histogram counts $(n_{\mathbf{r}}, \mathbf{r} \in \{1, \ldots, k_n\}^d)$. If we can find a differentially private version of $(n_{\mathbf{r}}, \mathbf{r} \in \{1, \ldots, k_n\}^d)$, then the corresponding density estimator $\hat{f}$ will also be differentially private by a simple change-of-measure argument. We consider the following perturbed histogram as described in [1]:

$$\hat{n}_{\mathbf{r}} = n_{\mathbf{r}} + z_{\mathbf{r}}, \forall \, \mathbf{r} \in \{1, \ldots, k_n\}^d, \tag{3}$$

where $z_{\mathbf{r}}$'s are independent with density $\alpha \exp(-\alpha|z|/2)/4$. We have

**Lemma 2** ([1]). $(\hat{n}_{\mathbf{r}}, \ \mathbf{r} \in \{1, \ldots, k_n\}^d)$ *satisfies $\alpha$-differential privacy.*

We call $(\hat{n}_{\mathbf{r}}, \ \mathbf{r} \in \{1, \ldots, k_n\}^d)$ the Perturbed Histogram. Substituting $n_{\mathbf{r}}$ by $\hat{n}_{\mathbf{r}}$ in (2), we obtain a differentially private version of $\hat{f}_{\text{hist}}$:

$$\hat{f}_{PH}(x) = h_n^{-d} \sum_{\mathbf{r}} \frac{\hat{n}_{\mathbf{r}}}{n} \mathbf{1}(x \in B_{\mathbf{r}}) . \tag{4}$$

In general $\hat{f}_{PH}$ given by (4) is not a valid density function, since it can take negative values and may not integrate to 1. To avoid these undesirable properties, [6] uses $\tilde{n}_{\mathbf{r}} = (\hat{n}_{\mathbf{r}} \vee 0)$ instead of $\hat{n}_{\mathbf{r}}$ and $\tilde{n} = \sum_{\mathbf{r}} \tilde{n}_{\mathbf{r}}$ instead of $n$ so that the resulting density estimator is non-negative and integrates to 1.

## 2.3 M-estimators

Given a random variable $X$ with density $f(x)$, the parameter of interest is defined as: $\theta^* = \arg\min_\Theta M(\theta)$, where $M(\theta) = \int m(x, \theta) f(x) dx$, $\Theta \subseteq \mathbf{R}^p$, and $m(x, \theta)$ is the *contrast function*. Assuming $X_i \overset{iid}{\sim} f$, the corresponding M-estimator is usually obtained by minimizing the empirical average of contrast function:

$$\hat{\theta} = \arg\min_{\theta \in \Theta} M_n(\theta), \text{ where } M_n(\theta) = n^{-1} \sum_{i=1} m(X_i, \theta). \tag{5}$$

M-estimators cover many important statistical inference procedures such as sample quantiles, maximum likelihood estimators (MLE), and least square estimators. Most M-estimators are $1/\sqrt{n}$-consistent and asymptotically normal. For more details about the theory and application of M-estimators, see [7].

## 3   Differentially private M-estimators

Combining equations (4) and (5) gives a differentially private objective function:

$$M_{n,PH}(\theta) = \int_{[0,1]^d} \hat{f}_{PH}(x) m(x, \theta) dx. \tag{6}$$

We wish to use the minimizer of $M_{n,PH}$ as a differentially private estimate of $\theta^*$. Consider the following set of conditions on the contrast function $m(x, \theta)$.

(A1)  $g(x, \theta) := \frac{\partial}{\partial \theta} m(x, \theta)$ exists and $|g(x, \theta)| \leq C_1$ on $[0, 1]^d \times \Theta$.

(A2)  $g(x, \theta)$ is Lipschitz in $x$ and $\theta$: $||g(x_1, \theta) - g(x_2, \theta)||_2 \leq C_2 ||x_1 - x_2||_2$, for all $\theta$; and $||g(x, \theta_1) - g(x, \theta_2)||_2 \leq C_2 ||\theta_1 - \theta_2||_2$, for all $x$.

(A3)  $m(x, \theta)$ is convex in $\theta$ for all $x$ and $M(\theta)$ is twice continuously differentiable with $M''(\theta^*) := \int f(x) \frac{\partial}{\partial \theta} g(x, \theta^*) dx$ positive definite.

Condition (A1) requires a bounded derivative of the contrast function, which is closely related to the *robustness* of the corresponding M-estimator [8]. It indicates that any small changes in the

underlying distribution cannot change the outcome by too much, which is also required implicitly by the definition of differential privacy. Condition (A2) has two parts. The Lipschitz condition on $x$ is used to bound the bias caused by histogram approximation, while the Lipschitz condition on $\theta$ is used to establish a uniform upper bound of the sampling error in $M_n'(\theta) = n^{-1} \sum_i g(x_i, \theta)$ as well as a uniform upper bound on the error caused by the additive Laplacian noises. Condition (A3) requires some curvature in the objective function in a neighborhood of the true parameter, which ensures that the minimizer is stable under small perturbations.

The following theorem is our first main result:

**Theorem 3.** *Under conditions (A1)-(A3), if $h_n \asymp (\sqrt{\log n}/n)^{2/(d+2)}$, then there exists a local minimizer, $\hat{\theta}_{PH}^*$, of $M_{n,PH}$, such that*

$$|\hat{\theta}_{PH}^* - \theta^*| = O_P\big(n^{-1/2} \vee (\sqrt{\log n}/n)^{2/(d+2)}\big). \tag{7}$$

A proof of Theorem 3 is given in the supplementary material. At a high level, by assumption (A3) it suffices to show (Lemma 9) that $\sup_{\theta \in \Theta_0} |M_{n,\text{priv}}'(\theta) - M'(\theta)| = O_P(1/\sqrt{n} \vee (\sqrt{\log n}/n)^{2/(2+d)})$, for some compact neighborhood $\Theta_0$ of $\theta^*$.

The approximation error of $M_{n,PH}'(\theta)$ can be decomposed into three parts:

$$
\begin{aligned}
\int (\hat{f}_{PH}(x) - f(x))g(x,\theta)dx =&\, n^{-1} \sum_{\mathbf{r}} z_{\mathbf{r}} h^{-d} \int_{B_{\mathbf{r}}} g(x,\theta)dx \\
&+ n^{-1} \sum_{\mathbf{r}} \left( n_{\mathbf{r}} h_n^{-d} \int_{B_{\mathbf{r}}} g(x,\theta)dx - \sum_{i:X_i \in B_{\mathbf{r}}}^{n} g(X_i,\theta) \right) \\
&+ n^{-1} \sum_i g(X_i,\theta) - Eg(X,\theta).
\end{aligned}
\tag{8}
$$

The three terms on the right hand side of (8) correspond to the effect of Laplace noises added for privacy, the bias caused by using histogram, and the sampling error, respectively. As in the general theory of histogram estimators, the approximation error depends on the choice of bandwidth $h_n$. Generally speaking, if the bandwidth is small, then the histogram bias term will be small. However, a smaller bandwidth leads to a larger number of cells and hence more Laplacian noises. As a result, there is a trade-off between the histogram bias and Laplacian noises in the choice of bandwidth. The bandwidth given in Theorem 3 balances these two parts. We also comment on practical choices of $h_n$ in Section 4.

We prove Theorem 3 by investigating the convergence rate of each term in the right hand side of (8). First (Lemma 10) by empirical process theory [9, 10] we have, under conditions A(1) and A(2), the sampling error term in (8) is of order $O_P(1/\sqrt{n})$, uniformly on $\Theta_0$. Second, using Lipschitz property of $g$, the histogram bias term in (8) is of order $O(h_n)$. Therefore it suffices to show that $\sup_{\theta \in \Theta_0} \big| \sum_{\mathbf{r}} n^{-1} z_{\mathbf{r}} h^{-d} \int_{B_{\mathbf{r}}} m(x,\theta)dx \big| = O_P\big((\sqrt{\log n}/n)h_n^{-d/2}\big)$, which can be established using a concentration inequality due to Talagrand [11] (see also [12, Equation 1.3]), together with a $\delta$-net argument (Lemma 11) enabled by the Lipschitz property of $g$ in $\theta$.

## 3.1 Algorithm based on perturbed histogram

In practice, exact integration of $\hat{f}_{PH}(x)m(x,\theta)$ over each cell $B_{\mathbf{r}}$ may be computationally expensive and approximations must be adopted to make the implementation feasible. Note that $\hat{f}_{PH}(x)$ is piecewise constant. The integration can be simplified by using a piecewise constant approximation of $m(x,\theta)$. Formally, we introduce the following algorithm:

**Algorithm 1 (M-estimator using perturbed histogram)**

Input: $D = \{X_1, \cdots, X_n\}$, $m(\cdot, \cdot)$, $\alpha$, $h_n$.

1. Construct perturbed histogram with bandwidth $h_n$ and privacy parameter $\alpha$ as in (3).
2. Let $M_{n,PH}(\theta) = n^{-1} \sum_{\mathbf{r}} \hat{n}_{\mathbf{r}} m(a_{\mathbf{r}}, \theta)$, where $a_{\mathbf{r}} \in [0,1]^d$ is the center of $B_{\mathbf{r}}$, with $a_{\mathbf{r}}(j) = (r_j - 0.5)h_n$ for all $1 \leq j \leq d$.

3. Output $\hat{\theta}_{PH} = \arg\min M_{n,PH}(\theta)$.

Comparing to $\hat{\theta}^*_{n,PH}$ obtained by minimizing the exact integral, the only term in (8) impacted by using $g(a_{\mathbf{r}}, \theta)$ instead of $h_n^{-d} \int_{B_{\mathbf{r}}} g(x, \theta)dx$ is the histogram bias term. However, note that

$$\left| g(a_{\mathbf{r}}, \theta) - h_n^{-d} \int_{B_{\mathbf{r}}} g(x, \theta)dx \right| = O(h_n).$$

As a result, the convergence rate of $\hat{\theta}_{n,PH}$ remains the same:

**Theorem 4** (Statistical Utility of Algorithm 1). *Under Assumptions (A1-A3), if $M_{n,PH}(\theta)$ is given by Algorithm 1 with $h_n \asymp (\sqrt{\log n}/n)^{2/(2+d)}$ then there exists a local minimizer, $\hat{\theta}_{PH}$, of $M_{n,PH}(\theta)$, such that*

$$|\hat{\theta}_{PH} - \theta^*| = O_P(1/\sqrt{n} \vee (\sqrt{\log n}/n)^{2/(2+d)}). \tag{9}$$

**Example** (Logistic regression) We give a concrete example that satisfies (A1)-(A3). Let $D = \{(X_i, Y_i) \in [0,1] \times \{0,1\} : 1 \leq i \leq n\}$, where the conditional distribution of $Y_i$ given $X_i$ is Bernoulli with parameter $\exp(\beta X_i)/[1 + \exp(\beta X_i)]$. The maximum likelihood estimator for $\beta$ is $\beta_{\mathrm{MLE}} = \arg\min \sum_i [-\beta Y_i X_i + \log(1 + \exp(\beta X_i))]$. Here the contrast function $m(x, y; \beta) = -\beta xy + \log(1 + \exp(\beta x))$ and it is easy to check that (A1)-(A3) hold. In this example $X$ is continuous and $Y$ is binary, so it is only necessary to discretize $X$ when constructing the histogram. To be specific, suppose $[0, 1]$ is partitioned into equal-sized cells $(B_r, 1 \leq r \leq k_n)$ as in the ordinary univariate histogram. The joint histogram for $(X, Y)$ is constructed by counting the number of data points in each of the product cells $B_{r,j} := B_r \times \{j\}$ for $j = 0, 1$. See Subsection 4.1 for more details on constructing histograms when there are categorical variables.

Note that Theorems 3 and 4 do not guarantee the uniqueness or even existence of a global minimizer for the perturbed objective function $M_{n,PH}(\theta)$. This is because sometimes with small probability some perturbed histogram count $\hat{n}_{\mathbf{r}}$ can be negative hence the corresponding objective function $M_{n,PH}$ may not be convex. In our simulation and real data experience, this is usually not a real problem since a similar argument as in Theorem 3 shows that, with high probability, the second derivative $M''_{n,PH}$ is uniformly close to $M''$ in any compact subset of $\Theta$. To completely avoid this issue, one can use thresholding after perturbation as described in the following algorithm.

**Algorithm 1′ (Perturbed histogram with nonnegative counts)**

Input: $D = \{X_1, \cdots, X_n\}$, $m(\cdot, \cdot)$, $\alpha$, $h_n$.

1 Construct perturbed histogram with bandwidth $h_n$ and privacy parameter $\alpha$ as in (3).

2 Let $\tilde{M}_{n,PH}(\theta) = n^{-1} \sum_{\mathbf{r}} \tilde{n}_{\mathbf{r}} m(a_{\mathbf{r}}, \theta)$, where $\tilde{n}_{\mathbf{r}} = \max(\hat{n}_{\mathbf{r}}, 0)$.

3 Output $\tilde{\theta}_{PH} = \arg\min \tilde{M}_{n,PH}(\theta)$.

Although the thresholding guarantees that the zero points of $M'_{n,PH}(\theta)$ is indeed a global minimizer by convexity of $M_{n,PH}(\theta)$, it increases the approximation error introduced by the Laplacian noises because now these noises no longer cancel with each other nicely in the first term of the right hand side of equation (8). We have the following utility result for Algorithm 1′:

**Theorem 5.** *Under Assumptions (A1-A3) and $h_n \asymp (\log n/n)^{1/(1+d)}$, the estimator given by Algorithm 1′ satisfies*

$$|\tilde{\theta}_{PH} - \theta^*| = O_P((\log n/n)^{1/(1+d)}).$$

*Proof.* The proof follows essentially from that of Theorem 3, with a different choice of bandwidth $h_n$. The concentration inequality result no longer holds for $\sum_{\mathbf{r}} \tilde{z}_{\mathbf{r}} g(a_{\mathbf{r}}, \theta)$ where $\tilde{z}_{\mathbf{r}} = \max(z_{\mathbf{r}}, -n_{\mathbf{r}})$, because $\tilde{z}_{\mathbf{r}}$'s are not independent. Instead, we consider a direct union bound: $\sup_{\mathbf{r}} |\tilde{z}_{\mathbf{r}}| \leq \sup_{\mathbf{r}} |z_{\mathbf{r}}| = O_P(\log h_n^{-d}) = O_P(\log n)$. Therefore the Laplacian noise term in right hand side of (8) is bounded uniformly for all $\theta$ by $O_P(n^{-1} h_n^{-d} \log n)$. The histogram bias is still $O(h_n)$ as we mentioned in the discussion of Algorithm 1. Therefore the convergence rate is optimized by choosing $h_n \asymp (\log n/n)^{1/(1+d)}$. $\qquad\square$

## 3.2 Non-differentiable contrast functions

Now we consider the possibility of relaxing condition (A2). Allowing discontinuity in $g(x, \theta)$ is motivated by a class of M-estimators whose contrast functions $m(x, \theta)$ are non-differentiable on a set of zero measure. An important example is the quantile. For a random variable $X \in \mathbf{R}^1$ with cumulative distribution function $F(\cdot)$ and any given $\tau \in (0, 1)$, the $\tau$-th quantile of $X$ is $q(\tau) := F^{-1}(\tau)$, which corresponds to an M-estimator with $m(x, \theta) = (1-\tau)(x-\theta)_- + \tau(x-\theta)_+$ (see [13]). Quantiles provide important information about the distribution, including both location (median) and scale (inter-quartile range). The robustness of sample quantiles also makes them good candidates for differentially private data release. Differentially private quantile estimators are indeed major building blocks for some existing privacy preserving statistical estimators [4, 5]. Our result in this subsection shows that perturbed histograms can give simple, consistent, and differentially private quantile estimators. The following set of conditions will suffice for this purpose and the argument is largely the same as Theorem 4:

(B1) $m(x, \theta)$ is convex and Lipschitz in both $x$ and $\theta$.

(B2) $M(\theta)$ is twice differentiable at $\theta^*$ with $M''(\theta^*) > 0$.

(B3) $\Theta$ is compact and convex.

**Corollary 6** (Statistical utility of Algorithm 1)**.** *Under conditions (B1-B3) and* $h_n \asymp (\sqrt{\log n}/n)^{2/(2+d)}$, *any minimizer* $\hat{\theta}_{PH}$ *of* $M_{n,PH}$ *given by Algorithm 1 satisfies (9).*

*Proof.* The argument is largely the same as the proof of Theorem 3. Here we consider the original objective functions $M_{n,PH}$ and $M$ instead of their derivatives. By a similar decomposition as in eq. (8), using the compactness of $\Theta$, we have $\sup_\Theta |M_{n,PH} - M| = O_P(1/\sqrt{n} \vee (\sqrt{\log n}/n)^{-2/(2+d)})$. Then the convergence of $\hat{\theta}_{PH}$ follows from the convexity of $M$. $\qquad\square$

*Remark* 7. Condition (B3) is the most restrictive one. It requires $\Theta$ to be bounded. This is because the proof uses the fact that $M_n(\theta)$ and $M(\theta)$ are uniformly close for large $n$, which is usually true for a bounded set of $\theta$.

*Remark* 8. For quantiles the contrast function is piecewise linear, so for most cells in the histogram there would be no approximation error if the data points are approximated by the cell center. The M-estimators for quantiles actually enjoy faster convergence rates.

**Extension to distributions supported on** $(-\infty, \infty)$. Recall that we assume $X \in [0, 1]^d$. For quantiles, we have $d = 1$ and the quantile estimators described above can be extended to any continuous random variable whose density function is supported on $(-\infty, \infty)$. Let $\{Z_i, i = 1, \ldots, n\}$ be an independent sample from density $f_Z$ with $f_Z(z) > 0$, $\forall z \in \mathbf{R}^1$. Let $\tau \in (0, 1)$ and suppose we want to estimate $q_Z(\tau)$, the $\tau$-th quantile of $Z$. To apply our method, define $X = \exp(Z)/(1 + \exp(Z))$. Clearly the quantiles are preserved under this monotone transformation. Applying the perturbed histogram quantile estimator on $\{X_i, i = 1, \ldots, n\}$ we obtain $\hat{q}_{X,PH}(\tau)$, the differentially private $\tau$-th qunatile of $X$, which is $1/\sqrt{n}$-consistent by Corollary 6. As a result, the estimate $\hat{q}_{Z,PH}(\tau) := \log[\hat{q}_{X,PH}(\tau)/(1 - \hat{q}_{X,PH}(\tau))]$ is a $1/\sqrt{n}$-consistent estimator for $q_Z(\tau)$.

## 4 Practical Aspects

### 4.1 Complexity and Flexibility

From now on we will drop the logarithm terms to simplify presentation. Suppose $h_n \asymp n^{-2/(2+d)}$. Then the perturbed histogram $(\hat{n}_\mathbf{r} : \mathbf{r} \in \{1, \ldots, h_n^{-1}\}^d)$ can be constructed in $O(n^{2d/(2+d)})$ time by specifying the corresponding cell for each data point. Once the histogram is constructed, following Algorithm 1, we can view it as a set of $h_n^{-d} = O(n^{2d/(2+d)})$ weighted data points $\{a_\mathbf{r}, \mathbf{r} \in \{1, \ldots, h_n^{-1}\}^d\}$ associated with weights $\{\hat{n}_\mathbf{r}\}$, where each data point $a_\mathbf{r}$ is the center of cell $B_\mathbf{r}$ as defined in Step 2 of Algorithm 1. For M-estimators that allow a close form solution in terms of the minimum sufficient statistics, such as least square regression, $M_{n,PH}(\theta)$ (and hence $\hat{\theta}_{PH}$) can be calculated in $O(n^{2d/(2+d)})$ time. For general M-estimators that require an iterative optimization, such as logistic regression, the Hessian and gradients can be calculated in $O(n^{2d/(2+d)})$

time in each iteration. Such a weighted sample representation can be easily implemented using standard data structures in common statistical programming packages such as `R` and `Matlab`.

Another attractive property of the proposed approach is its flexibility to accommodate different data types. As seen in the logistic regression example in Subsection 3.1, it is straightforward to construct multivariate histograms when some variables are categorical and some are continuous. In such cases it suffices to discretize the continuous variables. To be specific, let $(X^1, \ldots, X^{d_1}) \in [0,1]^{d_1}$ be a $d_1$-dimensional continuous variable and $(Y^1, \ldots, Y^{d_2}) \in \prod_{j=1}^{d_2}\{1, \ldots, k_j\}$ be a set of $d_2$ discrete variables where $Y^j$ takes value in $\{1, \ldots, k_j\}$. For any bandwidth $h$, let $\{B_{\mathbf{r}}, \mathbf{r} \in \{1, \ldots, h^{-1}\}^{d_1}\}$ be the corresponding set of histogram cells in $[0,1]^{d_1}$. Then the joint histogram for $(X, Y)$ is constructed with cells

$$\big\{B_{\mathbf{r},\mathbf{y}}, \mathbf{r} \in \{1, \ldots, h^{-1}\}^{d_1}, \mathbf{y} \in \bigotimes_{j=1}^{d_2}\{1, \ldots, k_j\}\big\}.$$

Because only the continuous variables have histogram approximation error, the theoretical results developed in Section 3 are applicable with sample size $n$ and dimensionality $d_1$.

## 4.2 Improvement by enhanced thresholding

In applications such as regression, the multivariate distribution often concentrates on a subset (usually a lower dimensional manifold) of $[0,1]^d$. Therefore many non-zero cells are artificially created by additive noises. To alleviate this problem, we threshold the histogram with an enhanced cut-off value: $\tilde{n}_{\mathbf{r}} = \hat{n}_{\mathbf{r}}\mathbf{1}(\hat{n}_{\mathbf{r}} \geq A\log n/\alpha)$, where $A > 0$ is a tuning parameter. This is based on the intuition that the maximal noise will be $O(\log n/\alpha)$. As shown in the following data example, such a simple thresholding step remarkably improves the accuracy.

## 4.3 Application to housing price data

As an illustration, we apply our method to a housing price data consisting of 348,189 houses sold in San Francisco Bay Area between 2003 and 2006. For each house, the data contains the price, size, year of transaction, and county in which the house is located. The inference problem of interest is to study the relationship between housing price and other variables [14]. In our case, we want to build a simple linear regression model to predict the housing price using the other variables while protecting each individual transaction record with differential privacy.

The data set has two continuous variables (price and size), one ordinal variable (year of sale) with 4 levels, and one categorical variable (county) with 9 levels. The preprocessing filters out data points with price outside of the range $\$10^5 \sim \$9 \times 10^5$ or with size larger than 3000 sqft. We also combine small counties that are geologically close and have similar housing prices. After the preprocessing, there are 250,070 data points and the county variable has 6 levels after the combination.

For each (year, county) combination, a perturbed histogram is constructed over the two continuous variables with privacy parameter $\alpha$ and $K$ levels in each continuous dimension. Then there are $4 \times 6 \times K^2$ cells, each having a perturbed histogram count. Using the weighted sample representation described in Subsection 4.1, the perturbed data can be viewed as a data set with $24K^2$ data points weighted by the perturbed histogram counts. A differentially private regression coefficient is obtained by applying a weighted least square regression on this data set. To assess the performance, the privacy preserving regression coefficients are compared with those given by the non-private ordinary least square (OLS) estimates. In particular, we look at the coordinate-wise relative deviance from OLS coefficients: $\varepsilon = |\hat{\theta}_{\mathrm{priv}}/\hat{\theta}_{\mathrm{OLS}} - 1|$. To account for the randomness of additive noises, we repeat 100 times and report the root mean square error: $\bar{\varepsilon} = (\sum_1^{100} \varepsilon_i^2/100)^{1/2}$, where $\varepsilon_i$ is the relative error obtained in the $i$th repetition. The results are summarized in Table 1.

We test 2 values of $\alpha$, the privacy parameter. Recall that a smaller value of $\alpha$ indicates a stronger privacy guarantee. For each value of $\alpha$ we apply both the original Algorithm 1 and the enhanced thresholding described in Subsection 4.2, with tuning parameter $A = 1/2$. For $\alpha = 1$ the coefficients given by the perturbed histogram are close to those given by OLS with most relative deviances below 5%. When $\alpha = 0.1$, which is a conservative choice because $\exp(0.1) \approx 1.1$, the perturbed histogram still gives reasonably close estimates with average deviance below 10% for all parameters

Table 1: Linear regression coefficients using the Bay Area housing data. The second column is the regression coefficients given by ordinary least square method without any perturbation. We compare estimate given by (1) perturbed histogram (PH, Algorithm 1) and (2) perturbed histogram with enhanced thresholding (THLD) as described in Subsection 4.2. The reported number is the root mean square relative error (in percentage) over 100 perturbations as described above. The histogram with use $K = 10$ segments in each continuous dimension.

| Variable | OLS | $\alpha = 0.1$ | | $\alpha = 1$ | |
|---|---|---|---|---|---|
| | | PH | THLD | PH | THLD |
| Intercept | 135141 | 10.6 | 7.7 | 7.2 | 4.4 |
| Size | 209 | 4.7 | 3.5 | 3.6 | 2.3 |
| Year | 56375 | 4.6 | 2.8 | 1.0 | 0.4 |
| County2 | -53765 | 8.0 | 7.8 | 1.5 | 0.7 |
| County3 | 146593 | 4.2 | 2.5 | 0.8 | 0.3 |
| County4 | -27546 | 29.8 | 37.1 | 2.8 | 2.1 |
| County5 | 45828 | 9.8 | 7.9 | 1.4 | 1.3 |
| County6 | -140738 | 7.1 | 3.3 | 1.0 | 0.4 |

except the county dummy variable "County4". This variable has the smallest OLS coefficient among all county dummy variables, so weight fluctuation in the histogram causes a relatively larger impact on the relative deviance. Even though, the perturbed histogram still gives at least qualitatively correct estimate. We also observe that the thresholded histogram gives more accurate estimate for all coefficients except for County4 when $\alpha = 0.1$.

The choice of $K$ should depend on the sample size and dimensionality. Our theory suggests $K = O(n^{2/(2+d)})$ where $d$ is the dimensionality of the histogram and hence equals the number of continuous variables. In this data set $n = 250,070$ and $d = 2$, which suggests $K \approx 500$. This is not a good choice since it produces $24 \times 500^2 = 6 \times 10^6$ cells. Let the number of cells be $c(K)$. In practice, it makes sense to choose $K$ such that the average data counts in a cell, $n/c(K)$, is much larger than the maximum additive noise $\max_{\mathbf{r}} |z_{\mathbf{r}}|$, which is $O_P(\log c(K))$. For this data set, when $K = 10$ we have $n/c(K) \approx 100$ and $\log(c(K)) \approx 7.78$.

## 5 Further Discussions

We demonstrate how histograms can be used as a basic tool for statistical parameter estimation under strong privacy constraints. The perturbed histogram adds to each histogram count a double-exponential noise with constant parameter depending only on the privacy budget $\alpha$. The histogram approximation bias and the additive noise on the cell counts result in a bias-variance trade-off as usually seen for histogram-based methods. Such an algorithm should work well for low-dimensional problems. Solutions to higher dimensional problems are yet to be developed. One possibility is to perturb the minimum sufficient statistics because the dimensionality of minimum sufficient statistics is usually much smaller than the number of histogram cells. For example, in linear regression analysis, it suffices to obtain the first and second moments of all variables in a privacy-preserving way. However, perturbing minimum sufficient statistics would only work for a single estimator and is only possible for interactive release. We are seeing another type of privacy-utility trade-off, where the utility is not only about the rate of convergence, but also about the range of possible analyses allowed by the data releasing mechanism.

The perturbed histogram is also related to "error in variable" inference problems. Suppose the original data is just the histogram, then the perturbed version can be thought as the true histogram counts contaminated by some measurement errors. In this paper we provide consistency results for a class of inference problems in presence of such measurement errors. However, plugging in the perturbed values does not necessarily give the best inference procedure and better alternatives may be possible, see [15] for a hypothesis testing example in contingency tables. An important and challenging question is how to find the optimal inference procedure in presence of such measurement errors. A positive answer to this question will help establish a lower bound of approximation error and better understand the power and limit of perturbed histograms.

**Acknowledgements**

Jing Lei was partially supported by NSF Grant BCS-0941518.

## Footnotes

[1]To make sure that $B_{\mathbf{r}}$'s do form a partition of $[0,1]^d$, the interval should be $[(k_n - 1)h_n, 1]$ when $r_j = k_n$.

# References

[1] C. Dwork, F. McSherry, K. Nissim, and A. Smith. Calibrating noise to sensitivity in private data analysis. In *Proceedings of the 3rd Theory of Cryptography Conference*, pages 265–284, 2006.

[2] C. Dwork. Differential privacy. In *Proceedings of the 33rd International Colloquium on Automata, Languages and Programming (ICALP)(2)*, pages 1–12, 2006.

[3] K. Chaudhuri and C. Monteleoni. Privacy-preserving logistic regression. In *Advances in Neural Information Processing Systems*, 2008.

[4] C. Dwork and J. Lei. Differential privacy and robust statistics. In *Proceedings of the 41st Annual ACM Symposium on Theory of Computing*, 2009.

[5] A. Smith. Privacy-preserving statistical estimation with optimal convergence rates. In *Proceedings of the 41st Annual ACM Symposium on Theory of Computing*, 2011.

[6] L. Wasserman and S. Zhou. A statistical framework for differential privacy. *Journal of the American Statistical Association*, 105:375–389, 2010.

[7] P. J. Huber and E. M. Ronchetti. *Robust Statistics*. John Wiley & Sons, Inc., 2nd edition, 2009.

[8] F. Hampel, E. Ronchetti, P. Rousseeuw, and W. Stahel. *Robust Statistics: The Approach Based on Influence Functions*. John Wiley, New York, 1986.

[9] A. W. van der Vaart. *Asymptotic Statistics*. Cambridge University Press, 1998.

[10] M. Talagrand. Sharper bounds for Gaussian and empirical processes. *The Annals of Probability*, 22:28–76, 1994.

[11] M. Talagrand. A new isoperimetric inequality and the concentration of measure phenomenon. *Lecture Notes in Mathematics*, 1469/1991:94–124, 1991.

[12] S. Bobkov and M. Ledoux. Poincaré's inequalities and Talagrand's concentration phenomenon for the exponential distribution. *Probability Theory and Related Fields*, 107:383–400, 1997.

[13] R. Koenker and K. F. Hallock. Quantile regression. *Journal of Economic Perspectives*, 15:143–156, 2001.

[14] R. K. Pace and R. Barry. Sparse spatial autoregressions. *Statistics & Probability Letters*, 33:291–297, 1997.

[15] D. Vu and A. Slavkovic. Differential privacy for clinical trial data: Preliminary evaluations. In *Proceedings of the 2009 IEEE International Conference on Data Mining Workshops*, 2009.

